# CONVERGENCE AND PATTERN STABILIZATION IN THE BOLTZMANN MACHINE

**Moshe Kam**
Dept. of Electrical and Computer Eng.
Drexel University, Philadelphia PA 19104

**Roger Cheng**
Dept. of Electrical Eng.
Princeton University, NJ 08544

## ABSTRACT

The Boltzmann Machine has been introduced as a means to perform global optimization for multimodal objective functions using the principles of simulated annealing. In this paper we consider its utility as a spurious-free content-addressable memory, and provide bounds on its performance in this context. We show how to exploit the machine's ability to escape local minima, in order to use it, at a constant temperature, for unambiguous associative pattern-retrieval in noisy environments. An association rule, which creates a *sphere of influence* around each stored pattern, is used along with the Machine's dynamics to match the machine's noisy input with one of the pre-stored patterns. Spurious fixed points, whose regions of attraction are not recognized by the rule, are skipped, due to the Machine's finite probability to escape from any state. The results apply to the Boltzmann machine and to the asynchronous net of binary threshold elements ('Hopfield model'). They provide the network designer with worst-case and best-case bounds for the network's performance, and allow polynomial-time tradeoff studies of design parameters.

## I. INTRODUCTION

The suggestion that artificial neural networks can be utilized for classification, pattern recognition and associative recall has been the main theme of numerous studies which appeared in the last decade (e.g. Rumelhart and McClelland (1986) and Grossberg (1988) - and their references.) Among the most popular families of neural networks are fully connected networks of binary threshold elements (e.g. Amari (1972), Hopfield (1982).) These structures, and the related family of fully connected networks of sigmoidal threshold elements have been used as error-correcting decoders in many applications, among which were interesting applications in optimization (Hopfield and Tank, 1985; Tank and Hopfield, 1986; Kennedy and Chua, 1987.) A common drawback of the many studied schemes is the abundance of 'spurious' local minima, which 'trap' the decoder in undesirable, and often non-interpretable, states during the process of input / stored-pattern association. It is generally accepted now that while the number of arbitrary binary patterns that can be stored in a fully-connected network is of the order of magnitude of N (N = number of the neurons in the network,) the number of created local attractors in the

network's state space is exponential in N.

It was proposed (Ackley et al., 1985; Hinton and Sejnowski, 1986) that fully-connected binary neural networks, which update their states on the basis of stochastic state-reassessment rules, could be used for *global* optimization when the objective function is multi-modal. The suggested architecture, the Boltzmann machine, is based on the principles of simulated annealing ( Kirkpatrick et al., 1983; Geman and Geman, 1984; Aarts et al., 1985; Szu, 1986,) and has been shown to perform interesting tasks of decision making and optimization. However, the learning algorithm that was proposed for the Machine, along with its "cooling" procedures, do not lend themselves to real-time operation. Most studies so far have concentrated on the properties of the Machine in global optimization and only few studies have mentioned possible utilization of the Machine (at constant 'temperature') as a content-addressable memory (e.g. for local optimization.)

In the present paper we propose to use the Boltzmann machine for *associative retrieval*, and develop bounds on its performance as a content-addressable memory. We introduce a learning algorithm for the Machine, which locally maximizes the stabilization probability of learned patterns. We then proceed to calculate (in polynomial time) upper and lower bounds on the probability that a tuple at a given initial Hamming distance from a stored pattern will get attracted to that pattern. A proposed association rule creates a *sphere of influence* around each stored pattern, and is indifferent to 'spurious' attractors. Due to the fact that the Machine has a nonzero probability of escape from any state, the 'spurious' attractors are ignored. The obtained bounds allow the assessment of retrieval probabilities, different learning algorithms and necessary learning periods, network 'temperatures' and coding schemes for items to be stored.

## II. THE MACHINE AS A CONTENT ADDRESSABLE MEMORY

The Boltzmann Machine is a multi-connected network of N simple processors called *probabilistic binary neurons*. The $i^{th}$ neuron is characterized by N-1 real numbers representing the *synaptic weights* ($w_{ij}$, j=1,2,...,i-1,i+1,....,N; $w_{ii}$ is assumed to be zero for all i), a real *threshold* ($\tau_i$) and a binary *activity level* ($u_i \in B = \{-1,1\}$,) which we shall also refer to as the *neuron's state*. The binary N-tuple $U = [u_1,u_2,. . . ,u_N]$ is called the *network's state*. We distinguish between two phases of the network's operation:

a) The learning phase - when the network parameters $w_{ij}$ and $\tau_i$ are determined. This determination could be achieved through autonomous learning of the binary pattern environment by the network (unsupervised learning); through learning of the environment with the help of a 'teacher' which supplies evaluative reinforcement signals (supervised learning); or by an external fixed assignment of parameter values.

b) The production phase - when the network's state U is determined. This determination could be performed synchronously by all neurons at the same predetermined time instants, or asynchronously - each neuron reassesses its state independently of the other neurons at random times. The decisions of the neurons regarding their states during reassessment can be arrived at deterministically (the set of neuron inputs determines the neuron's state) or

stochastically (the set of neuron inputs shapes a probability distribution for the state-selection law.)

We shall describe first the (asynchronous and stochastic) production rule which our network employs. At random times during the production phase, asynchronously and independently of all other neurons, the $i^{th}$ neuron decides upon its next state, using the probabilistic decision rule

$$u_i = \begin{cases} 1 & \text{with probabilty } \dfrac{1}{1+e^{-\frac{\Delta E_i}{T_e}}} \\[4em] -1 & \text{with probabilty } \dfrac{e^{-\frac{\Delta E_i}{T_e}}}{1+e^{-\frac{\Delta E_i}{T_e}}} \quad \text{where} \quad \Delta E_i = \sum_{j=1, j\neq i}^{N} w_{ij}u_j - \tau_i \end{cases} \tag{1}$$

is called the $i^{th}$ *energy gap*, and $T_e$ is a predetermined real number called *temperature*.

The state-updating rule (1) is related to the network's *energy level* which is described by

$$E = -\frac{1}{2}\left[ \sum_{i=1}^{N} u_i \left( \sum_{j=1, j\neq i}^{N} w_{ij}u_j - \tau_i \right) \right]. \tag{2}$$

If the network is to find a local minimum of E in equation (2), then the $i^{th}$ neuron, when chosen (at random) for state updating, should choose deterministically

$$u_i = \text{sgn}\left[ \sum_{j=1, j\neq i}^{N} w_{ij}u_j - \tau_i \right]. \tag{3}$$

We note that rule in equation (3) is obtained from rule in equation (1) as $T_e \to 0$. This deterministic choice of $u_i$ guarantees, under symmetry conditions on the weights ($w_{ij}=w_{ji}$), that the network's state will stabilize at a *fixed point* in the $2^N$-tuple state space of the network (Hopfield, 1982), where

Definition 1: A state $U_f \in B^N$ is called a *fixed point* in the state space of the N-neuron network if

$$P_r[U^{(n_p+1)} = U_f \mid U^{(n_p)} = U_f] = 1. \tag{4}$$

A fixed point found through iterations of equation (3) (with i chosen at random at each iteration) may not be the *global* minimum of the energy in equation (2). A mechanism which seeks the global minimum should avoid local-minimum "traps" by allowing 'uphill' climbing with respect to the value of *E*. The decision scheme of equation (1) is devised for that purpose, allowing an increase in *E* with nonzero probability. This provision for progress in the locally 'wrong' direction in order to reap a 'global' advantage later, is in accordance with the principles of *simulated annealing* techniques, which are used in multimodal optimization. In our case, the probabilities of choosing the locally 'right' decision (equation (3)) and the locally 'wrong' decision are determined by the ratio of the energy gap $\Delta E_i$ to the 'temperature' shaping constant $T_e$ .

The Boltzmann Machine has been proposed for global minimization and a considerable amount of effort has been invested in devising a good *cooling scheme*, namely a means to control $T_e$ in order to guarantee the finding of a global minimum in a short time (Geman and Geman, 1984, Szu, 1987.) However, the network may also be used as a selective *content addressable memory* which does not suffer from inadvertently-installed spurious local minima.

We consider the following application of the Boltzmann Machine as a scheme for pattern classification under noisy conditions: let an encoder emit a sequence of NX1 binary code vectors from a set of Q codewords (or 'patterns',) each having a probability of occurrence of $\Pi_m$ (m = 1,2,. . . ,Q). The emitted pattern encounters noise and distortion before it arrives at the decoder, resulting in some of its bits being reversed. The Boltzmann Machine, which is the decoder at the receiving end, accepts this distorted pattern as its initial state ($U^{(0)}$), and observes the consequent time evolution of the network's state U. At a certain time instant $n_0$, the Machine will declare that the input pattern $U^{(0)}$ is to be associated with pattern $B_m$ if U at that instant ( $U^{(n_0)}$) is 'close enough' to $B_m$. For this purpose we define

Definition 2: The $d_{max}$-*sphere of influence of pattern* $B_m$, $\sigma( B_m, d_{max})$ is

$$\sigma(B_m, d_{max})= \{U \in B^N : HD(U, B_m) \le d_{max}\}. \tag{5}$$

$d_{max}$ is prespecified.

Let $\Sigma(d_{max})= \cup_m \sigma(B_m, d_{max})$ and let $n_0$ be the smallest integer such that $U^{(n_0)} \in \Sigma(d_{max})$.

The **rule of association** is : associate $U^{(0)}$ with $B_m$ at time $n_0$, if $U^{(n_0)}$ which has evolved

from $U^{(0)}$ satisfies: $U^{(n_0)} \in \sigma(B_m, d_{max})$.

Due to the finite probability of escape from any minimum, the energy minima which correspond to spurious fixed points are skipped by the network on its way to the energy valleys induced by the designed fixed points (i.e. $B_1,. . . ,B_Q$.)

## III. ONE-STEP CONTRACTION PROBABILITIES

Using the association rule, the utility of the Boltzmann machine for error correction involves the probabilities

$$P_r \{HD [U^{(n)},B_m] \le d_{max} \mid HD [U^{(0)},B_m]= d\} \quad m=1,2, \ldots, Q \tag{6}$$

for predetermined $n$ and $d_{max}$ . In order to calculate (6) we shall first calculate the following one-step *attraction probabilities* :

$$P(B_m,d,\delta)= P_r\{HD[U^{(n_p + 1)}, B_m]=d+\delta \mid HD[U^{(n_p)}, B_m]=d\} \text{ where } \delta = -1, 0, 1 \tag{7}$$

For $\delta$ = -1 we obtain the *probability of convergence* ; For $\delta$ = +1 we obtain the *probability of divergence* ; For $\delta$ = 0 we obtain the *probability of stalemate*.

An exact calculation of the attraction probabilities in equation (7) is time-exponential and we shall therefore seek lower and upper bounds which can be calculated in polynomial time. We shall reorder the weights of each neuron according to their contribution to the

$\Delta E_i$ for each pattern, using the notation

$$W_i^m = \{w_{i1}b_{m1}, w_{i2}b_{m2}, \ldots, w_{iN}b_{mN}\} \qquad \Omega_{i1}^m = \max W_i^m$$

$$\Omega_{is}^m = \max\{ W_i^m - \{\Omega_{i1}^m, \Omega_{i2}^m, \ldots, \Omega_{is-1}^m\} \} \qquad (8)$$

$$i = 1,2,\ldots,N,\ s = 2,3,\ldots,N,\ m = 1,2,\ldots,Q$$

$$\text{Let } \Delta E_{mi}^-(d) = \Delta E_{mi} - 2\sum_{r=1}^{d} \Omega_{ir}^m \text{ and } \Delta E_{mi}^+(d) = \Delta E_{mi} - 2\sum_{r=1}^{d} \Omega_{iN+1-r}^m \qquad (9)$$

These values represent the maximum and minimum values that the $i^{th}$ energy gap could assume when the network is at HD of d from $B_m$. Using these extrema, we can find the *worst case attraction probabilities* :

$$P^{wc}(B_m,d,-1) = \frac{d}{N}\sum_{i=1}^{N}\rho_i\left[\frac{U_{-1}(b_{mi})}{1+e^{-\frac{\Delta E_m^-(d)}{T_e}}} + \frac{1-U_{-1}(b_{mi})}{1+e^{-\frac{\Delta E_m^+(d)}{T_e}}}e^{-\frac{\Delta E_m^+(d)}{T_e}}\right] \quad (10a)$$

$$P^{wc}(B_m,d,1) = \frac{N-d}{N}\sum_{i=1}^{N}\rho_i\left[\frac{U_{-1}(b_{mi})e^{-\frac{\Delta E_m^-(d)}{T_e}}}{1+e^{-\frac{\Delta E_m^-(d)}{T_e}}} + \frac{1-U_{-1}(b_{mi})}{1+e^{-\frac{\Delta E_m^+(d)}{T_e}}}\right] \quad (10b)$$

and the *best case attraction probabilities* :

$$P^{bc}(B_m,d,-1) = \frac{d}{N}\sum_{i=1}^{N}\rho_i\left[\frac{U_{-1}(b_{mi})}{1+e^{-\frac{\Delta E_m^+(d)}{T_e}}} + \frac{1-U_{-1}(b_{mi})}{1+e^{-\frac{\Delta E_m^-(d)}{T_e}}}e^{-\frac{\Delta E_m^-(d)}{T_e}}\right] \quad (11a)$$

$$P^{bc}(B_m,d,1) = \frac{N-d}{N}\sum_{i=1}^{N}\rho_i\left[\frac{U_{-1}(b_{mi})e^{-\frac{\Delta E_m^+(d)}{T_e}}}{1+e^{-\frac{\Delta E_m^+(d)}{T_e}}} + \frac{1-U_{-1}(b_{mi})}{1+e^{-\frac{\Delta E_m^-(d)}{T_e}}}\right] \quad (11b)$$

where for both cases
$$P^{wc(bc)}(B_m,d,0) = 1 - P^{wc(bc)}(B_m,d,-1) - P^{wc(bc)}(B_m,d,1). \qquad (12)$$

For the worst- (respectively, best-) case probabilities, we have used the extreme values of $\Delta E_{mi}(d)$ to underestimate (overestimate) convergence and overestimate (underestimate) divergence, given that there is a disagreement in d of the N positions between the network's state and $B_m$; we assume that errors are equally likely at each one of the bits.

## IV. ESTIMATION OF RETRIEVAL PROBABILITIES

To estimate the retrieval probabilities, we shall study the Hamming distance of the

network's state from a stored pattern. The evolution of the network from state to state, as affecting the distance from a stored pattern, can be interpreted in terms of a birth-and-death Markov process (e.g. Howard, 1971) with the probability transition matrix

$$
\Psi_{bd}(p_{bi}, p_{di}) =
\begin{bmatrix}
1-p_{b0} & p_{b0} & 0 & 0 & 0 & & & & 0 & 0 \\
p_{d1} & 1-p_{b1}-p_{d1} & p_{b1} & 0 & 0 & & & & & 0 \\
0 & p_{d2} & 1-p_{b2}-p_{d2} & p_{b2} & 0 & & & & & 0 \\
0 & 0 & & & & & & & & 0 \\
\cdot & & \cdot & & & & & & & \\
\cdot & & \cdot & & 0 & p_{dk} & 1-p_{bk}-p_{dk} & p_{bk} & 0 & \\
\cdot & & \cdot & & & & & & & \\
0 & & & & & & 0 & & & \\
0 & & & & & & 0 & p_{dN-1} & 1-p_{bN-1}-p_{dN-1} & p_{bN-1} \\
0 & & & & & & 0 & 0 & p_{dN} & 1-p_{dN}
\end{bmatrix}
$$

$$(13)$$

where the birth probability $p_{bi}$ is the divergence probability of increasing the HD from i to i+1, and the death probability $p_{di}$ is the contraction probability of decreasing the HD from i to i-1.

Given that an input pattern was at HD of $d_0$ from $B_m$, the probability that after n steps the network will associate it with $B_m$ is

$$
P_r\{U^{(n)} \to B_m] \mid HD[U^{(0)}, B_m] = d_0\} = \sum_{r=0}^{d_{max}} P_r[HD(U^{(n)}, B_m) = r \mid HD(U^{(0)}, B_m) = d_0] \quad (14)
$$

where we can use the one-step bounds found in section III in order to calculate the worst-case and best-case probabilities of association. Using equations (10) and (11) we define two matrices for each pattern $B_m$; a worst case matrix, $\Psi_{bd}^{wc}$, and a best case matrix, $\Psi_{bd}^{bc}$:

**Worst case matrix**

$$P_{bi} = P^{wc}(B_m, i, +1)$$

$$P_{di} = P^{wc}(B_m, i, -1)$$

**Best case matrix**

$$P_{bi} = P^{bc}(B_m, i, +1)$$

$$P_{di} = P^{bc}(B_m, i, -1).$$

Using these matrices, it is now possible to calculate lower and upper bounds for the association probabilities needed in equation (14):

$$
[\pi_{d_0}(\Psi_{bd}^{wc})^n]_r \leq P_r[HD(U^{(n)}, B_m) = r \mid HD(U^{(0)}, B_m) = d_0] \leq [\pi_{d_0}(\Psi_{bd}^{bc})^n]_r \quad (15a)
$$

where $[x]_i$ indicates the $i^{th}$ element of the vector x, and $\pi_{d_0}$ is the unit 1 x n+1 vector

$$[\pi_{d_0}]_i = \begin{cases} 1 & i = d_0 \\ \\ 0 & i \neq d_0 \end{cases} \qquad (15b)$$

The bounds of equation 14(a) can be used to bound the association probability in equation (13). The *upper bound* of the association probability is obtained by replacing the summed terms in (13) by their upper-bound values:

$$P_r\{U^{(n)} \rightarrow B_m] \mid HD[U^{(0)}, B_m] = d_0\} \leq \sum_{r=0}^{d_{max}} [\pi_{d_0}(\Psi_{bd}^{bc})^n]_r \qquad (16a)$$

The *lower bound* cannot be treated similarly, since it is possible that at some instant of time prior to the present time-step (n), the network has already associated its state U with one of the other patterns. We shall therefore use as the lower bound on the convergence probability in equation (14):

$$\sum_{r=0}^{d_{max}} [\pi_{d_0}(\underline{\Psi_{bd}^{wc}})^n]_r \leq P_r\{U^{(n)} \rightarrow B_m] \mid HD[U^{(0)}, B_m]\} \qquad (16b)$$

where the underlined matrix is the birth-and-death matrix (13) with

$$P_{bi} = \begin{cases} P^{wc}(B_m, i, +1) \\ \\ 1 \end{cases} \qquad P_{di} = \begin{cases} P^{wc}(B_m, i, -1) & \text{for } i = 1, 2, \ldots, \mu_i - 1 \\ \\ 0 & \text{for } i = \mu_i, \mu_i + 1, \ldots, N \end{cases} \qquad (16c)$$

and

$$\mu_i = \min HD(B_i, B_j) - d_{max} \quad j = 1, \ldots, Q, j \neq i \qquad (16d)$$

Equation (16c) and (16d) assume that the network wanders into the $d_{max}$- *sphere of influence* of a pattern other than $B_i$, whenever its distance from $B_i$ is $\mu_i$ or more. This assumption is very conservative, since $\mu_i$ represents the shortest distance to a competing $d_{max}$- *sphere of influence*, and the network could actually wander to distances larger than $\mu_i$ and still converge eventually into the $d_{max}$-*sphere of influence* of $B_i$.

## CONCLUSION

We have presented how the Boltzmann Machine can be used as a content-addressable memory, exploiting the stochastic nature of its state-selection procedure in order to escape undesirable minima. An association rule in terms of patterns' *spheres of influence* is used, along with the Machine's dynamics, in order to match an input tuple with one of the predetermined stored patterns. The system is therefore indifferent to 'spurious' states, whose spheres of influence are not recognized in the retrieval process. We have detailed a technique to calculate the upper and lower bounds on retrieval probabilities of each stored

pattern. These bounds are functions of the network's parameters (i.e. assignment or learning rules, and the pattern sets); the initial Hamming distance from the stored pattern; the association rule; and the number of production steps. They allow a polynomial-time assessment of the network's capabilities as an associative memory for a given set of patterns; a comparison of different coding schemes for patterns to be stored and retrieved; an assessment of the length of the learning period necessary in order to guarantee a prespecified probability of retrieval; and a comparison of different learning/assignment rules for the network parameters. Examples and additional results are provided in a companion paper (Kam and Cheng, 1989).

**Acknowledgements**

This work was supported by NSF grant IRI 8810186.

**References**

[1]    Aarts,E.H.L., Van Laarhoven,P.J.M. : "Statistical Cooling: A General Approach to Combinatorial Optimization Problems," *Phillips J. Res.*, Vol. 40, 1985.
[2]    Ackley,D.H., Hinton,J.E., Sejnowski,T.J. : " A Learning Algorithm for Boltzmann Machines," *Cognitive Science*, Vol. 19, pp. 147-169, 1985.
[3] Amari,S-I : "Learning Patterns and Pattern Sequences by Self-Organizing Nets of Threshold Elements," *IEEE Trans. Computers*, Vol. C-21, No. 11, pp. 1197-1206, 1972.
[4]    Geman,S., Geman,D. : "Stochastic Relaxation, Gibbs Distributions, and the Bayesian Restoration of Images" *IEEE Trans. Patt. Anal. Mach. Int.*, pp. 721-741, 1984.
[5]    Grossberg,S. : "Nonlinear Neural Networks: Principles, Mechanisms, and Architectures," *Neural Networks*, Vol. 1, 1988.
[6]    Hebb,D.O. : *The Organization of Behavior*, New York:Wiley, 1949.
[7]    Hinton,J.E., Sejnowski,T.J.   " Learning and Relearning in the Boltzmann Machine," in [14]
[8]    Hopfield,J.J. : "Neural Networks and Physical Systems with Emergent Collective Computational Abilities," *Proc. Nat. Acad. Sci. USA*, pp. 2554-2558, 1982.
[9]    Hopfield,J.J., Tank,D. :" 'Neural' Computation of Decision in Optimization Problems," *Biological Cybernetics*, Vol. 52, pp. 1-12, 1985.
[10] Howard,R.A.: *Dynamic Probabilistic Systems*, New York:Wiley, 1971.
[11] Kam,M., Cheng,R.: " Decision Making with the Boltzmann Machine," *Proceedings of the 1989 American Control Conference*, Vol. 1, Pittsburgh, PA, 1989.
[12] Kennedy,M.P., Chua, L.O. :"Circuit Theoretic Solutions for Neural Networks," *Proceedings of the 1st Int. Conf. on Neural Networks*, San Diego, CA, 1987.
[13] Kirkpatrick,S., Gellat,C.D.,Jr., Vecchi,M.P. : "Optimization by Simulated Annealing," *Science*, 220, pp. 671-680, 1983.
[14]    Rumelhart,D.E., McClelland,J.L. (editors): *Parallel Distributed Processing, Volume 1: Foundations*, Cambridge:MIT press, 1986.
[15] Szu,H.: "Fast Simulated Annealing," in Denker,J.S.(editor) : *Neural Networks for Computing*, New York:American Inst. Phys., Vol. 51.,pp. 420-425, 1986.
[16] Tank,D.W., Hopfield, J.J. : "Simple 'Neural' Optimization Networks," *IEEE Transactions on Circuits and Systems*, Vol. CAS-33, No. 5, pp. 533-541, 1986.
